# Balanced Graph Matching

**Timothee Cour,    Praveen Srinivasan    and Jianbo Shi**
Department of Computer and Information Science
University of Pennsylvania
Philadelphia, PA 19104
{timothee,psrin,jshi}@seas.upenn.edu

## Abstract

Graph matching is a fundamental problem in Computer Vision and Machine Learning. We present two contributions. First, we give a new spectral relaxation technique for approximate solutions to matching problems, that naturally incorporates one-to-one or one-to-many constraints within the relaxation scheme. The second is a normalization procedure for existing graph matching scoring functions that can dramatically improve the matching accuracy. It is based on a reinterpretation of the graph matching compatibility matrix as a bipartite graph on edges for which we seek a bistochastic normalization. We evaluate our two contributions on a comprehensive test set of random graph matching problems, as well as on image correspondence problem. Our normalization procedure can be used to improve the performance of many existing graph matching algorithms, including spectral matching, graduated assignment and semidefinite programming.

## 1   Introduction

Many problems of interest in Computer Vision and Machine Learning can be formulated as a problem of correspondence: finding a mapping between one set of points and another set of points. Because these point sets can have important internal structure, they are often considered not simply as point sets, but as two separate graphs. As a result, the correspondence problem is commonly referred to as graph matching. In this setting, graph nodes represent feature points extracted from each instance (e.g. a test image and a template image) and graph edges represent relationships between feature points. The problem of graph matching is to find a mapping between the two node sets that preserves as much as possible the relationships between nodes.

Because of its combinatorial nature, graph matching is either solved exactly in a very restricted setting (bipartite matching, for example with the Hungarian method) or approximately. Most of the recent literature on graph matching has followed this second path, developing approximate relaxations to the graph matching problem. In this paper, we make two contributions. The first contribution is a spectral relaxation for the graph matching problem that incorporates one-to-one or one-to-many mapping constraints, represented as affine constraints. A new mathematical tool is developed for that respect, Affinely Constrained Rayleigh Quotients. Our method achieves comparable performance to state of the art algorithms, while offering much better scalability. Our second contribution relates to the graph matching scoring function itself, which we argue, is prone to systematic confusion errors. We show how a proper bistochastic normalization of the graph matching compatibility matrix is able to considerably reduce those errors and improve the overall matching performance. This improvement is demonstrated both for our spectral relaxation algorithm, and for three state of the art graph matching algorithms: spectral matching, graduated assignment and semidefinite programming.

## 2 Problem formulation

**Attributed Graph**   We define an attributed graph[1] as a graph $G = (V, E, A)$ where each edge $e = ij \in E$ is assigned an attribute $A_e$, which could be a real number or a vector in case of multi-attributes. We represent vertex attributes as special edge attributes, i.e. $A_{ii}$ for a vertex $i$. For example, the nodes could represent feature points with attributes for spatial location/orientation and image feature descriptors, while edge attributes could represent spatial relationships between two nodes such as relative position/orientation.

**Graph Matching Cost**   Let $G = (V, E, A)$, $G' = (V', E', A')$ be two attributed graphs. We want to find a mapping between $V$ and $V'$ that best preserves the attributes between edges $e = ij \in E$ and $e' = i'j' \in E'$. Equivalently, we seek a set of correspondences, or matches $M = \{ii'\}$ so as to maximize the graph matching score, defined as:

$$\epsilon_{GM}(M) \quad = \sum_{ii' \in M, jj' \in M} f(A_{ij}, A'_{i'j'}) = \sum_{e \sim e'} f(A_e, A'_{e'}), \tag{1}$$

with the shorthand notation $e \sim e'$ iff $ii' \in M, jj' \in M$. The function $f(\cdot, \cdot)$ measures the similarity between edge attributes. As a special case, $f(A_{ii}, A'_{i'i'})$ is simply the score associated with the match $ii'$. In the rest of the paper, we let $n = |V|$, $m = |E|$, and likewise for $n', m'$.

**Formulation as Integer Quadratic Program**   We explain here how to rewrite (1) in a more manageable form. Let us represent $M$ as a binary vector $x \in \{0,1\}^{nn'}$: $x_{ii'} = 1$ iff $ii' \in M$. For most problems, one requires the matching to have a special structure, such as one-to-one or one-to-many: this is the *mapping constraint*. For one-to-one matching, this is $\sum_{i'} x_{ii'} = 1$ and $\sum_{i} x_{ii'} = 1$ (with $x$ binary), and $M$ is a permutation matrix. In general, this is an affine inequality constraint of the form $Cx \leq b$. With those notations, (1) takes the form of an Integer Quadratic Program (IQP):

$$\max \quad \epsilon(x) = x^\mathsf{T} W x \quad \text{s.t.} \quad Cx \leq b, \quad x \in \{0,1\}^{nn'} \tag{2}$$

$W$ is a $nn' \times nn'$ compatibility matrix with $W_{ii', jj'} = f(A_{ij}, A'_{i'j'})$. In general such IQP is NP-hard, and approximate solutions are needed.

**Graph Matching Relaxations**   Continuous relaxations of the IQP (2) are among the most successful methods for non-bipartite graph matching, and so we focus on them. We review three state of the art matching algorithms: semidefinite programming (SDP) [2, 3], graduated assignment (GA) [4], and spectral matching (SM) [5]. We also introduce a new method, Spectral Matching with Affine Constraints (SMAC) that provides a tigher relaxation than SM (and more accurate results in our experiments) while still retaining the speed and scalability benefits of spectral methods, which we also quantify in our evaluations. All of these methods relax the original IQP into a continuous program (removing the $x \in \{0,1\}$ constraint), so we omit this step in the derivations below.

**SDP Relaxation**   In [2], the authors rewrite the objective as a matrix innner product: $x^\mathsf{T} W x = \langle X, W_{eq} \rangle$, where $X = [1; x]^\mathsf{T} [1; x]$ is a $(nn' + 1) \times (nn' + 1)$ rank-one matrix and $W_{eq} = \begin{bmatrix} 0 & d^\mathsf{T}/2 \\ d/2 & W - D \end{bmatrix}$, where $d = \text{diag}(W)$ and $D$ is a diagonal matrix of diagonal $d$. The non-convex rank-one constraint is further relaxed by only requiring $X$ to be positive semi-definite. Finally the relaxation is: $\max \quad \langle X, W_{eq} \rangle \quad \text{s.t.} \quad \langle X, C_{eq}^{(i)} \rangle \leq b_{eq}^{(i)}, X \succeq 0$, for suitable $C_{eq}, b_{eq}$. The relaxation squares the problem size, which we will see, prevents SDP from scaling to large problems.

**Graduated Assignment**   GA[4] relaxes the IQP into a non-convex quadratic program (QP) by removing the constraint $x \in \{0,1\}$. It then solves a sequence of convex approximations, each time by maximizing a Taylor expansion of the QP around the previous approximate solution. The accuracy of the approximation is controlled by a continuation parameter, annealed after each iteration.

**Spectral Matching (SM)**   In [5], the authors drop the constraint $Cx \leq b$ during relaxation and only incorporate it during the discretization step. The resulting program: $\max \quad x^\mathsf{T} W x \quad \text{s.t.} \quad ||x|| = 1$, which is the same as $\max \quad \frac{x^\mathsf{T} W x}{x^\mathsf{T} x}$, can be solved by computing the leading eigenvector $x$ of $W$. It verifies $x \geq 0$ when $W$ is nonnegative, by Perron-Frobenius' theorem.

# 3  Spectral Matching with Affine Constraint (SMAC)

We present here **our first contribution, SMAC**. Our method is closely related to the spectral matching formulation of [5], but we are able to impose affine constraints $Cx = b$ on the relaxed solution. We demonstrate later that the ability to maintain this constraint, coupled with scalability and speed of spectral methods, results in a very effective solution to graph matching. We solve the following:

$$\max \quad \frac{x^\mathsf{T} W x}{x^\mathsf{T} x} \quad \text{s.t.} \quad Cx = b \qquad (3)$$

Note, for one-to-one matching the objective *coincides with the IQP* for binary $x$ since $x^\mathsf{T} x = n$.

**Computational Solution**   We can formulate (3) as maximization of a *Rayleigh quotient under affine constraint*. While the case of *linear* constraints has been addressed previously[6], imposing affine constraints is novel. We fully address this class of problem in the *supplementary material*[1] and give a brief summary here. The solution to (3) is given by the leading eigenpair of

$$P_C W P_C \quad x = \lambda x, \qquad (4)$$

where $x$ is scaled so that $Cx = b$ exactly. We introduced $P_C = I_{nn'} - C_{eq}^T (C_{eq} C_{eq}^T)^{-1} C_{eq}$ and $C_{eq} = [I_{k-1}, 0] (C - (1/b_k) b C_k)$, where $C_k, b_k$ denote the last row of $C, b$ and $k = \#$ constraints.

**Discretization**   We show here how to tighten our approximation during the discretization step in the case of one-to-one matching (we can fall back to this case by introducing dummy nodes). Let us assume for a moment that $n = n'$. It is a well known result that for any $n \times n$ matrix $X$, $X$ is a *permutation* matrix *iff* $X1 = X^\mathsf{T} 1 = 1$, $X$ is orthogonal, and $X \geq 0$ elementwise. We show here we can obtain a tighter relaxation by *incorporating the first 2 (out of 3) constraints* as a post-processing before the final discretization. We carry on the following steps even when $n \neq n'$: **1)** reshape the solution $x$ of (3) into a $n \times n'$ matrix $X$, **2)** compute the best orthogonal approximation $X_{orth}$ of $X$. It can be computed using the SVD decomposition $X = U\Sigma V^T$, similarly to [7]: $X_{orth} = \arg\min \{||X - Q|| : Q \in O(n, n')\} = UV^T$, where $O(n, n')$ denotes the orthogonal matrices of $\mathbb{R}^{n \times n'}$, and **3)** discretize $X_{orth}$ like the other methods, as explained in the results section. The following proposition shows $X_{orth}$ is orthogonal and satisfies the affine constraint, as promised.

**Proposition 3.1 ($X_{orth}$ satisfies the affine constraint)**  *If* $\mathbf{u}$ *is left* **and** *right eigenvector of a matrix* $Y$, *then* $\mathbf{u}$ *is left and right eigenvector of* $Y_{orth}$. *Corollary: when* $n = n'$, $X_{orth} 1 = X_{orth}{}^\mathsf{T} 1 = 1$.

**Proof**: see supplementary materials. Note that in general, $X$ and $X_{orth}$ do *not* have the same eigenvectors, here we are lucky because of the particular constraint induced by $C, b$.

**Computational Cost**   The cost of this algorithm is dominated by the computation of the leading eigenvector of (4), which is function of two terms: 1) number of matrix-vector operations required in an eigensolver (which we can fix, as convergence is fast in practice), and 2) cost per matrix-vector operation. $P_C$ is a full matrix, even when $C$ is sparse, but we showed the operation $y := P_C x$ can be computed in $O(nn')$ using the Sherman-Morrison formula (for one-to-one matching). Finally, the total complexity is proportional to the number of non-zero elements in $W$. If we assume a full-matching, this is $O(mm')$, which is *linear* in the problem description length.

# 4  How Robust is the Matching?

We ran extensive graph matching experiments on both real image graphs and synthetic graphs with the algorithms presented above. We noticed a clear trend: the algorithms get confused when there is ambiguity in the compatibility matrix. Figure 1 shows a typical example of what happens. We extracted a set of feature points (indexed by $i$ and $i'$) in two airplane images, and for each edge $e = ij$ in the first graph, we plotted the most similar edges $e' = i'j'$ in the second graph. As we can see, the first edge plotted has many correspondences everywhere in the image and is therefore *uninformative*. The second edge on the other hand has correspondences with roughly only 5 locations, it is *informative*, and yet its contribution is outweighed by the first edge. The compatibility matrix is *unbalanced*. We illustrate next what happens with a synthetic example.

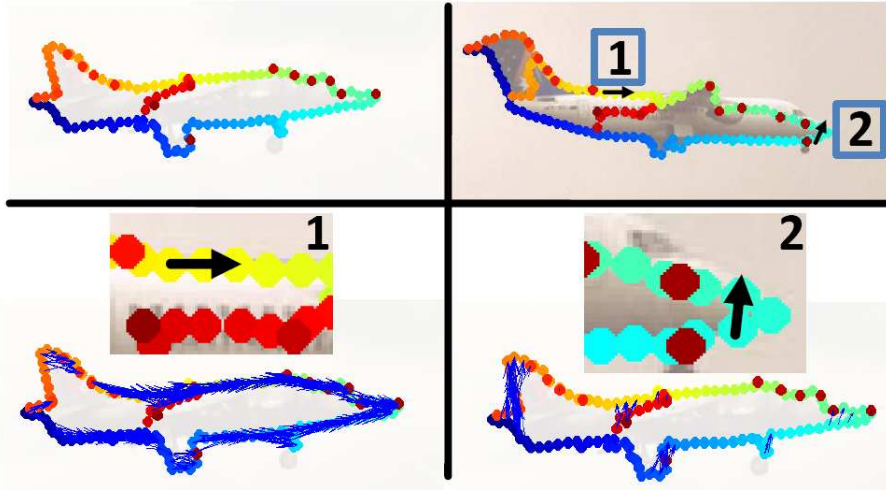

Figure 1: Representative cliques for graph matching. Blue arrows indicate edges with high similarity, showing 2 groups: cliques of type 1 (pairing roughly horizontal edges in the 2 images) are *uninformative*, cliques of type 2 (pairing vertical edges) are *distinctive*.

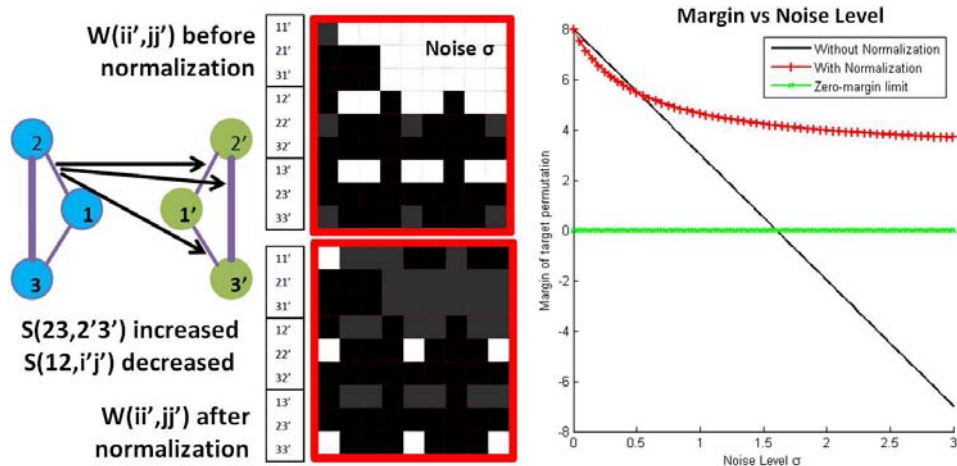

Figure 2: Left: edges 12 and 13 are *uninformative* and make spurious connections of strength $\sigma$ to all edges in the second graph. Edge 23 is *informative* and makes a single connection to the second graph, 2'3'. Middle: corresponding compatibility matrices $W$ (top: before normalization, bottom: after normalization). Right: margin as a function of $\sigma$ (difference between correct matching score and best runner-up score).

**Synthetic noise model example**  Let us look at a synthetic example to illustrate this concept, on which the IQP can be solved by brute-force. Figure 2 shows two isomorphic graphs with 3 nodes. In our simple noise model, edges 12 and 13 are *uninformative* and make connections to every edge in the second graph, with strength $\sigma$ (our noise parameter). The *informative* edge 23 on the other hand only connects to $2'3'$. We displayed $W_{ii',jj'}$ to visualize the connections. When the noise is small enough, the optimal matching is the desired permutation $\mathbf{p}^* = \{11', 22', 33'\}$, with an initial score of 8 for $\sigma = 0$. We computed the score of the second best permutation as a function of $\sigma$ (see plot of margin), and showed that for $\sigma$ greater than $\sigma_0 \approx 1.6$, $\mathbf{p}^*$ is no longer optimal. $W$ is *unbalanced*, with some edges making spurious connections, overwhelming the influence of other edges with few connections. This problem is not incidental. In fact we argue this is the main source of confusion for graph matching. The next section introduces a normalization algorithm to address this problem.

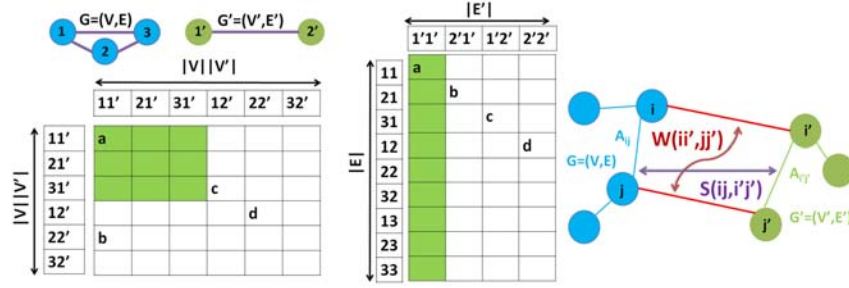

Figure 3: Left: matching compatibility matrix $W$ and edge similarity matrix $S$. The shaded areas in each matrix correspond to the same entries. Right: graphical representation of $S, W$ as a clique potential on $i, i', j, j'$.

## 5   How to balance the Compatibility Matrix

As we saw in the previous section, a main source of confusion for graph matching algorithms is the unbalance in the compatibility matrix. This confusion occurs when an edge $e \in E$ has many good potential matches $e' \in E'$. Such an edge is *not discriminative* and its influence should be *decreased*. On the other hand, an edge with *small number of good matches* will help disambiguate the optimal matching. Its influence should be *increased*. The following presents **our second contribution, bistochastic normalization**.

### 5.1   Dual Representation: Matching Compatibility Matrix $W$ vs. Edge Similarity Matrix $S$

The similarity function $f(\cdot, \cdot)$ can be interpreted in two ways: either as a *similarity* between edges $ij \in E$ and $i'j' \in E'$, or as a *compatibility* between match hypothesis $ii' \in M$ and $jj' \in M$. We define the **similarity matrix** $S$ of size $m \times m'$ as $S_{ij,i'j'} = f(A_{ij}, A'_{i'j'})$, and (as before) the **compatibility matrix** $W$ of size $nn' \times nn'$ as $W_{ii',jj'} = f(A_{ij}, A'_{i'j'})$, see Figure 3. Each vertex $i$ in the first graph should ideally match to a small number of vertices $i'$ in the second graph. Similarly, each edge $e = ij \in E$ should also match to a small number of edges $e' = i'j' \in E'$. Although this constraint would be very hard to enforce, we approach this behavior by normalizing the influence of each edge. This corresponds to having each row and column in $S$ (not $W$!) sum to one, in other words, $S$ *should be bistochastic*.

### 5.2   Bistochastic Normalization of Edge Similarity Matrix $S$

Recall we are given a compatibility matrix $W$. Can we enforce its dual representation $S$ to be bistochastic? One problem is that, even though $W$ is square (of size $nn' \times nn'$), $S$ could be rectangular (of size $m \times m'$), in which case its rows and columns cannot both sum to 1. We define a $m \times m'$ matrix $B$ to be **Rectangular Bistochastic** if it satisfies: $B1_{m'} = 1_m$ and $B^\mathsf{T}1_m = (m/m')1_{m'}$. We can formulate the normalization as solving the following balancing problem:

$$\text{Find } (D, D') \text{ diagonal matrices of order } m, m' \quad \text{s.t.} \quad DSD' \text{ is } \textit{rectangular bistochastic} \tag{5}$$

We propose the following algorithm to solve (5), and then show its correctness.

1. Input: compatibility matrix W, of size $nn' \times nn'$

2. Convert $W$ to $S$: $\quad S_{ij,i'j'} = W_{ii',jj'}$

3. repeat until convergence

   (a) normalize the rows of $S$: $\quad S^{t+1}_{ij,i'j'} := S^t_{ij,i'j'} / \sum_{k'l'} S^t_{ij,k'l'}$

   (b) normalize the columns of $S$: $\quad S^{t+2}_{ij,i'j'} := S^{t+1}_{ij,i'j'} / \sum_{kl} S^{t+1}_{kl,i'j'}$

4. Convert back $S$ to $W$, output $W$

**Proposition 5.1 (Existence and Uniqueness of (D,D'))** *Under the condition $S > 0$ elementwise, Problem (5) has a unique solution $(D, D')$, up to a scale factor. $D$ and $D'$ can be found by iteratively normalizing the rows and columns of $S$.*

**Proof** Let $\bar{S} = S \otimes 1_{m' \times m}$, which is square. Since $\bar{S} > 0$ elementwise, we can apply an existing version of (5.1) for square matrices[8]. We conclude the proof by noticing that normalizing rows and columns of $\bar{S}$ preserves kronecker structure: $\bar{D}\bar{S}\bar{D}' = (D \otimes 1_{m' \times m'})(S \otimes 1_{m' \times m})(D' \otimes 1_{m \times m}) = mm'DSD' \otimes 1_{m' \times m}$, and so $(m^2 D, m'D')$ is solution for $S$ iff $(\bar{D}, \bar{D}')$ is solution for $\bar{S}$ $\square$

We illustrate in Figure 2 the improvement of normalization on our previous synthetic example of noise model. Spurious correspondences are suppressed and informative correspondances such as $W_{23,2'3'}$ are enhanced, which makes the correct correspondence clearer. The plot on the right shows that normalization makes the matching robust to arbitrarily large noise in this model, while unnormalized correspondences will eventually result in incorrect matchings.

# 6 Experimental Results

**Discretization and Implementation Details** Because all of the methods described are continuous relaxations, a post-processing step is needed to discretize the continuous solution while satisfying the desired constraints. Given an initial solution estimate, GA finds a near-discrete local minimum of the IQP by solving a series of Taylor approximations. We can therefore use GA as follows: 1) initialize GA with the relaxed solution of each algorithm, and 2) discretize the output of GA with a simple greedy procedure described in [5]. **Software:** For SDP, we used the popular SeDuMi [9] optimization package. Spectral matching and SMAC were implemented using the standard Lanczos eigensolver available with MATLAB, and we implemented an optimized version of GA in C++.

## 6.1 One-to-one Attributed Graph Matching on Random Graphs

Following [4], we performed a comprehensive evaluation of the 4 algorithms on random one-to-one graph matching problems. For each matching problem, we constructed a graph $G$ with $n = 20$ nodes, and $m$ random edges ($m = 10\% n^2$ in a first series of experiments). Each edge $ij \in E$, was assigned a random attribute $A_{ij}$ distributed uniformly in $[0, 1]$. We then created a perturbed graph $G'$ by adding noise on the edge attributes: $A'_{i'j'} = A_{p(i)p(j)} + noise$, where $p$ is a random permutation; the noise was distributed uniformly in $[0, \sigma]$, $\sigma$ varying from 0 to 6. The compatibility matrix $W$ was computed from graphs $G, G'$ as follows: $W_{ii',jj'} = \exp(-|A_{ij} - A'_{i'j'}|^2), \forall ij \in E, i'j' \in E'$. For each noise level we generated 100 different matching problems and computed the average error rate by comparing the discretized matching to the ground truth permutation.

**Effect of Normalization on each method** We computed the average error rates with and without normalization of the compatibility matrix $W$, for each method: SDP, GA, SM and SMAC, see Figure 4. We can see dramatic improvement due to normalization, regardless of the relaxation method used. At higher noise levels, all methods had a 2 to 3-fold decrease in error rate.

**Comparison Across Methods** We plotted the performance of all 4 methods using *normalized* compatibility matrices in Figure 5 (left), again, with 100 trials per noise level. We can see that SDP and SMAC give comparable performance, while GA and especially SM do worse. These results validate SMAC with normalization as a state-of-the-art relaxation method for graph matching.

**Influence of edge density and graph size** We experimented with varying edge density: noise $\sigma = 2$, $n = 20$, edge density varying from 10% to 100% by increments of 10% with 20 trials per increment. For SMAC, the normalization resulted in an average absolute error reduction of 60%, and for all density levels the reduction was at least 40%. For SDP, the respective figures were 31%, 20%. We also did the same experiments, but with fixed edge density and varying graph sizes, from 10 to 100 nodes. For SMAC, normalization resulted in an average absolute error reduction of 52%; for all graph sizes the reduction was at least 40%.

**Scalability and Speed** In addition to accuracy, scalability and speed of the methods are also important considerations. Matching problems arising from images and other sensor data (e.g., range

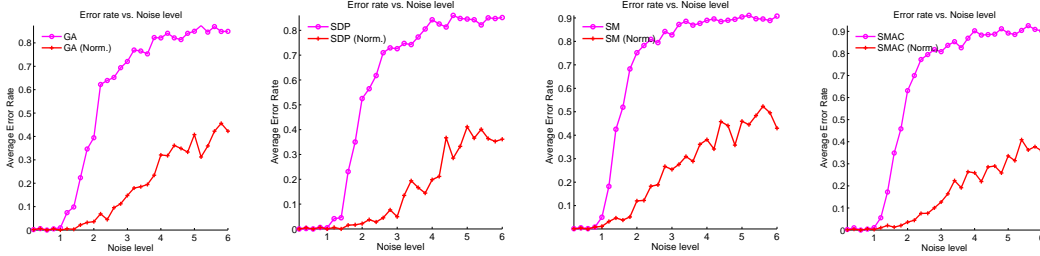

Figure 4: Comparison of matching performance with normalized and unnormalized compatibility matrices. Axes are error rate vs. noise level. In all cases (GA, SDP, SM, SMAC), the error rate decreases substantially.

scans) may have hundreds of nodes in each graph. As mentioned previously, the SDP relaxation squares the problem size (in addition to requiring expensive solvers), greatly impacting its speed and scalability. Figure 5 (middle and right) demonstrates this. For a set of random one-to-one matching problems of varying size $n$ (horizontal axis), we averaged the *time for computing the relaxed solution* of all four methods (10 trials for each $n$). We can see that SDP scales quite poorly (almost 30 minutes for $n = 30$). In addition, on a machine with 2GB of RAM, SDP typically ran out of memory for $n = 60$. By contrast, SMAC and SM scale easily to much larger problems ($n = 200$).

## 6.2 Image correspondence

We also tested the effect of normalization on a simple but instructive image correspondence task. In each of two images to match, we formed a multiple attribute graph by sub-sampling $n = 100$ canny edge points as graph nodes. Each pair of feature points $e = ij$ within 30 pixels was assigned two attributes: angle $\angle e = \angle \overrightarrow{ij}$ and distance $d(e) = ||\overrightarrow{ij}||$. $S$ was computed as follows: $S(e, e') = 1$ *iff* $\cos(\angle e' - \angle e) > \cos \pi/8$ and $\frac{|d(e) - d(e')|}{\min(d(e), d(e'))} < 0.5$. By using simple geometric attributes, we emphasized the effect of normalization on the energy function, rather than feature design.

Figure 6 shows an image correspondence example between the two airplane images of Figure 1. We display the result of SMAC with and without normalization. Correspondence is represented by similarly colored dots. Clearly, normalization improved the correspondence result. Without normalization, large systematic errors are made, such as mapping the bottom of one plane to the top of the other. With normalization these errors are largely eliminated.

Let us return to Figure 1 to see the effect of normalization on $S(e, e')$. As we saw, there are roughly 2 types of connections: 1) horizontal edges (uninformative) and 2) vertical edges (discriminative). Normalization exploits this disparity to enhance the latter edges: before normalization, each connection contributed up to $1.0$ to the overall matching score. After normalization, connections of type 2 contributed up to 0.64 to the overall matching score, versus 0.08 for connections of type 1, which is 8 times more. We can view normalization as imposing an upper bound on the contribution of each connection: the upper bound is smaller for spurious matches, and higher for discriminative matches.

## 7 Conclusion

While recent literature mostly focuses on improving relaxation methods for graph matching problems, we contribute both an improved relaxation algorithm, SMAC, and a method for improving the energy function itself, graph balancing with bistochastic normalization. In our experiments, SMAC outperformed GA and SM, with similar accuracy to SDP, it also scaled much better than SDP. We motivate the normalization with an intuitive example, showing it improves noise tolerance by enhancing informative matches and de-enhancing uninformative matches. The experiments we performed on random one-to-one matchings show that normalization dramatically improves both our relaxation method SMAC, and the three algorithms mentioned. We also demonstrated the value of normalization for establishing one-to-one correspondences between image pairs. Normalization imposes an upper bound on the score contribution of each edge in proportion to its saliency.

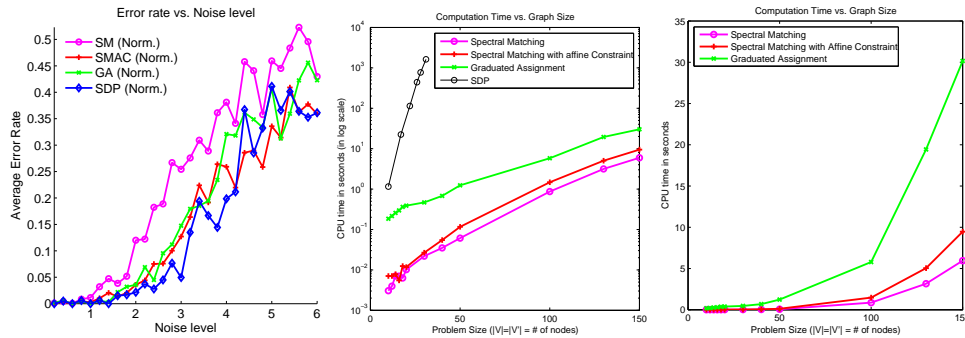

Figure 5: Left: comparison of different methods with **normalized compatibility matrices**. Axes: vertical is error rate averaged over 100 trials; horizontal is noise level. SMAC achieves comparable performance to SDP. Middle,right: computation times of graph matching methods (left: log-scale, right: linear scale).

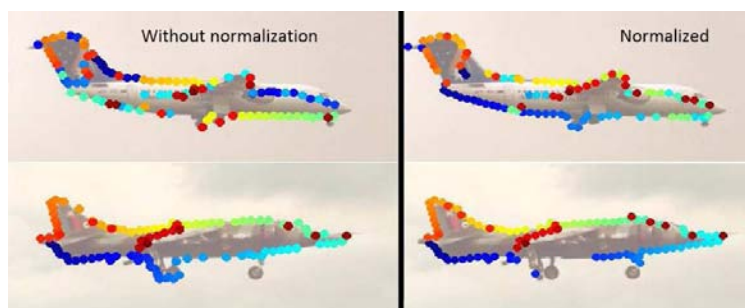

Figure 6: Image correspondence via SMAC with and without normalization; like colors indicate matches.

## Footnotes

[1]http://www.seas.upenn.edu/~timothee/

# References

[1] Marcello Pelillo. A unifying framework for relational structure matching. *icpr*, 02:1316, 1998.

[2] Christian Schellewald and Christoph Schnörr. Probabilistic subgraph matching based on convex relaxation. In *Energy Minimization Methods in Computer Vision and Pattern Recognition*, 2005.

[3] P.H.S. Torr. Solving markov random fields using semi definite programming. In *Artificial Intelligence and Statistics*, 2003.

[4] S. Gold and A. Rangarajan. A graduated assignment algorithm for graph matching. In *IEEE Transactions on Pattern Analysis and Machine Intelligence*, volume 18, 1996.

[5] Marius Leordeanu and Martial Hebert. A spectral technique for correspondence problems using pairwise constraints. In *International Conference on Computer Vision*, October 2005.

[6] Stella X. Yu and Jianbo Shi. Grouping with bias. In *Advances in Neural Information Processing Systems*, 2001.

[7] G.L.Scott and H.C.Longuett-Higgins. An algorithm for associating the features of two images. In *Proceedings of the Royal Society of London B*, 1991.

[8] Paul Knopp and Richard Sinkhorn. Concerning nonnegative matrices and doubly stochastic matrices. *Pacific J. Math*, 2:343–348, 1967.

[9] J.F. Sturm. Using SeDuMi 1.02, a MATLAB toolbox for optimization over symmetric cones. *Optimization Methods and Software*, 11–12:625–653, 1999. Special issue on Interior Point Methods.
